# Optimal Stochastic Search and Adaptive Momentum

**Todd K. Leen and Genevieve B. Orr**
Oregon Graduate Institute of Science and Technology
Department of Computer Science and Engineering
P.O.Box 91000, Portland, Oregon 97291-1000

## Abstract

Stochastic optimization algorithms typically use learning rate schedules that behave asymptotically as $\mu(t) = \mu_0/t$. The ensemble dynamics (Leen and Moody, 1993) for such algorithms provides an easy path to results on mean squared weight error and asymptotic normality. We apply this approach to stochastic gradient algorithms with momentum. We show that at late times, learning is governed by an *effective* learning rate $\mu_{eff} = \mu_0/(1 - \beta)$ where $\beta$ is the momentum parameter. We describe the behavior of the asymptotic weight error and give conditions on $\mu_{eff}$ that insure optimal convergence speed. Finally, we use the results to develop an adaptive form of momentum that achieves optimal convergence speed *independent* of $\mu_0$.

## 1 Introduction

The rate of convergence for gradient descent algorithms, both batch and stochastic, can be improved by including in the weight update a "momentum" term proportional to the previous weight update. Several authors (Tugay and Tanik, 1989; Shynk and Roy, 1988) give conditions for convergence of the mean and covariance of the weight vector for momentum LMS with *constant learning rate*. However stochastic algorithms require that the learning rate decay over time in order to achieve true convergence of the weight (in probability, in mean square, or with probability one).

This paper uses our previous work on weight space probabilities (Leen and Moody, 1993; Orr and Leen, 1993) to study the convergence of stochastic gradient algorithms with annealed learning rates of the form $\mu = \mu_0/t$, both with and without momentum. The approach provides simple derivations of previously known results and their extension to stochastic descent with momentum. Specifically, we show that the mean squared weight misadjustment drops off at the maximal rate $\propto 1/t$ only if the effective learning rate $\mu_{eff} = \mu_0/(1 - \beta)$ is greater than a critical value which is determined by the Hessian.

These results suggest a new algorithm that automatically adjusts the momentum coefficient to achieve the optimal convergence rate. This algorithm is simpler than previous approaches that either estimate the curvature directly during the descent (Venter, 1967) or measure an auxilliary statistic not directly involved in the optimization (Darken and Moody, 1992).

## 2   Density Evolution and Asymptotics

We consider stochastic optimization algorithms with weight $\omega \in R^N$. We confine attention to a neighborhood of a local optimum $\omega_*$ and express the dynamics in terms of the *weight error* $v \equiv \omega - \omega_*$. For simplicity we treat the continuous time algorithm [1]

$$\frac{dv(t)}{dt} = \mu(t) H[v(t), x(t)] \qquad (1)$$

where $\mu(t)$ is the learning rate at time $t$, $H$ is the weight update function and $x(t)$ is the data fed to the algorithm at time $t$. For stochastic gradient algorithms $H = -\nabla_v \mathcal{E}(v, x(t))$, minus the gradient of the instantaneous cost function.

Convergence (in mean square) to $\omega_*$ is characterized by the average squared norm of the weight error $E[|v|^2] = \text{Trace } C$ where

$$C \equiv \int d^N v \; v \, v^T \, P(v, t) \qquad (2)$$

is the weight error correlation matrix and $P(v, t)$ is the probability density at $v$ and time $t$. In (Leen and Moody, 1993) we show that the probability density evolves according to the Kramers-Moyal expansion

$$\frac{\partial P(v, t)}{\partial t} =$$

$$\sum_{i=1}^{\infty} \frac{(-1)^i}{i!} \sum_{j_1, \ldots j_i = 1}^{N} \frac{\partial^i}{\partial v_{j_1} \partial v_{j_2} \ldots \partial v_{j_i}} \left\{ \langle \mu H_{j_1} \, \mu H_{j_2} \ldots \mu H_{j_i} \rangle_x \, P(v, t) \right\}, \qquad (3)$$

where $H_{j_k}$ denotes the $j_k^{th}$ component of the $N$-component vector $H$, and $\langle \ldots \rangle_x$ denotes averaging over the density of inputs. Differentiating (2) with respect to time, using (3) and integrating by parts, we obtain the equation of motion for the weight error correlation

$$\frac{dC}{dt} = \mu(t) \int d^N v \, P(v,t) \left[ v \left\langle H(v,x)^T \right\rangle_x + \left\langle H(v,x) \right\rangle_x v^T \right] +$$

$$\mu(t)^2 \int d^N v \, P(v,t) \left\langle H(v,x) H(v,x)^T \right\rangle_x . \tag{4}$$

## 2.1  Asymptotics of the Weight Error Correlation

Convergence of $v$ can be understood by studying the late time behavior of (4). Since the update function $H(v,x)$ is in general non-linear in $v$, the time evolution of the correlation matrix $C_{ij}$ is coupled to higher moments $E[v_i v_j v_k \ldots]$ of the weight error. However, the learning rate is assumed to follow a schedule $\mu(t)$ that satisfies the requirements for convergence in mean square to a local optimum. Thus at late times the density becomes sharply peaked about $v = 0^2$. This suggests that we expand $H(v,x)$ in a power series about $v = 0$ and retain the lowest order non-trivial terms in (4) leaving:

$$\frac{dC}{dt} = -\mu(t) \left[ (R\,C) + (C\,R^T) \right] + \mu(t)^2 \, D , \tag{5}$$

where $R$ is the Hessian of the average cost function $\langle \mathcal{E} \rangle_x$, and

$$D \equiv \left\langle H(0,x) H(0,x)^T \right\rangle_x \tag{6}$$

is the diffusion matrix, both evaluated at the local optimum $\omega_*$. (Note that $R^T = R$.) We use (5) with the understanding that it is valid for large $t$. The solution to (5) is

$$C(t) = U(t,t_0) \, C(t_0) \, U^T(t,t_0) + \int_{t_0}^t d\tau \, \mu(\tau)^2 \, U(t,\tau) \, D \, U^T(t,\tau) . \tag{7}$$

where the evolution operator $U(t_2, t_1)$ is

$$U(t_2, t_1) = \exp \left[ -R \int_{t_1}^{t_2} d\tau \, \mu(\tau) \right] . \tag{8}$$

We assume, without loss of generality, that the coordinates are chosen so that $R$ is diagonal ($D$ won't be) with eigenvalues $\lambda_i$, $i = 1 \ldots N$. Then with $\mu(t) = \mu_0/t$ we obtain

$$E[\,|v|^2\,] = \text{Trace}\,[\,C(t)\,] = \sum_{i=1}^N \left\{ C_{ii}(t_0) \left( \frac{t_0}{t} \right)^{2\mu_0 \lambda_i} \right.$$

$$\left. + \frac{\mu_0^2 \, D_{ii}}{(2\,\mu_0\,\lambda_i - 1)} \left[ \frac{1}{t} - \frac{1}{t_0} \left( \frac{t_0}{t} \right)^{2\mu_0 \lambda_i} \right] \right\} . \tag{9}$$

We define

$$\mu_{crit} \equiv \frac{1}{2\,\lambda_{min}} \qquad (10)$$

and identify two regimes for which the behavior of (9) is fundamentally different:

1. $\mu_0 > \mu_{crit}$: $E[|v|^2]$ drops off asymptotically as $1/t$.

2. $\mu_0 < \mu_{crit}$: $E[|v|^2]$ drops off asymptotically as $\left(\frac{1}{t}\right)^{(2\,\mu_0\,\lambda_{min})}$
   i.e. *more slowly than* $1/t$.

Figure 1 shows results from simulations of an ensemble of 2000 networks trained by LMS, and the prediction from (9). For the simulations, input data were drawn from a gaussian with zero mean and variance $R = 1.0$. The targets were generated by a noisy teacher neuron (i.e. targets $=\omega_* x + \xi$, where $\langle \xi \rangle = 0$ and $\langle \xi^2 \rangle = \sigma^2$). The upper two curves in each plot (dotted) depict the behavior for $\mu_0 < \mu_{crit} = 0.5$. The remaining curves (solid) show the behavior for $\mu_0 > \mu_{crit}$.

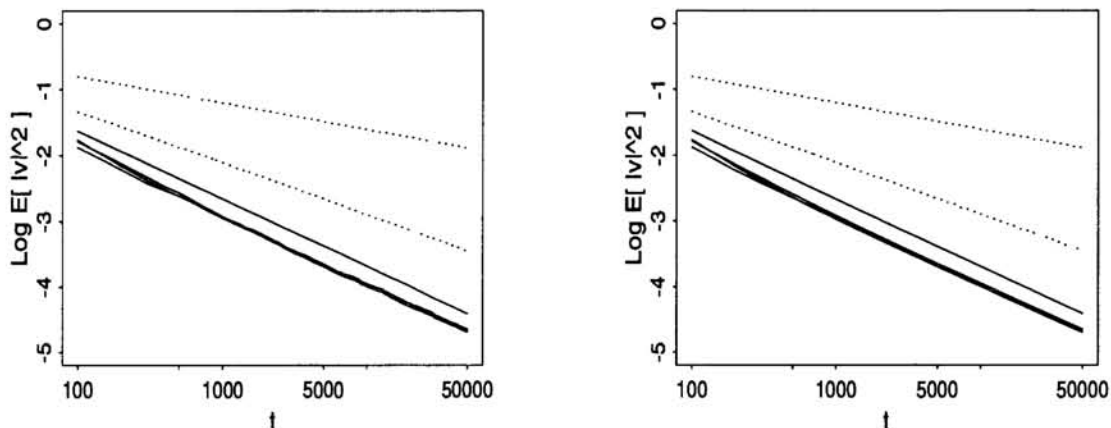

Fig.1: LEFT – Simulation results from an ensemble of 2000 one-dimensional LMS algorithms with $R = 1.0$, $\sigma^2 = 1.0$ and $\mu = \mu_0/t$. RIGHT – Theoretical predictions from equation (9). Curves correspond to (top to bottom) $\mu_0 = 0.2, 0.4, 0.6, 0.8, 1.0, 1.5$.

By minimizing the coefficient of $1/t$ in (9), the optimal learning rate is found to be $\mu_{opt} = 1/\lambda_{min}$. This formalism also yields asymptotic normality rather simply (Orr and Leen, 1994). These conditions for "optimal" (i.e. $1/t$) convergence of the weight error correlation and the related results on asymptotic normality have been previously discussed in the stochastic approximation literature (Darken and Moody, 1992; Goldstein, 1987; White, 1989; and references therein). The present formal structure provides the results with relative ease and facilitates the extension to stochastic gradient descent with momentum.

## 3   Stochastic Search with Constant Momentum

The discrete time algorithm for stochastic optimization with momentum is:

$$v(t+1) \quad = \quad v(t) + \mu(t)\,H[v(t), x(t)] + \beta\,\Omega(t) \qquad (11)$$

$$\begin{aligned}
\Omega(t+1) &\equiv v(t+1) - v(t) \\
&= \Omega(t) + \mu(t)\, H[v(t), x(t)] + (\beta - 1)\, \Omega(t),
\end{aligned} \tag{12}$$

or in continuous time,

$$\frac{dv(t)}{dt} = \mu(t)\, H[v(t), x(t)] + \beta\, \Omega(t) \tag{13}$$

$$\frac{d\Omega(t)}{dt} = \mu(t)\, H[v(t), x(t)] + (\beta - 1)\, \Omega(t). \tag{14}$$

As before, we are interested in the late time behavior of $E[\,|v|^2\,]$. To this end, we define the $2N$-dimensional variable $Z \equiv (v, \Omega)^T$ and, following the arguments of the previous sections, expand $H[v(t), x(t)]$ in a power series about $v = 0$ retaining the lowest order non-trivial terms. In this approximation the correlation matrix $\overline{C} \equiv E[ZZ^T]$ evolves according to

$$\frac{d\overline{C}}{dt} = K\overline{C} + \overline{C}K^T + \mu(t)^2\, \overline{D} \tag{15}$$

with

$$K \equiv \begin{pmatrix} -\mu(t)\, R & \beta I \\ -\mu(t) R & (\beta - 1) I \end{pmatrix}, \quad \overline{D} \equiv \begin{pmatrix} D & D \\ D & D \end{pmatrix}, \tag{16}$$

$I$ is the $N \times N$ identity matrix, and $R$ and $D$ are defined as before. The evolution operator is now

$$\overline{U}(t_2, t_1) \equiv \exp\left[ \int_{t_1}^{t_2} d\tau\, K(\tau) \right] \tag{17}$$

and the solution to (15) is

$$\overline{C} = \overline{U}(t, t_0)\, \overline{C}(t_0)\, \overline{U}^T(t, t_0) + \int_{t_0}^{t} d\tau\, \mu^2(\tau)\, \overline{U}(t, \tau)\, \overline{D}\, \overline{U}^T(t, \tau) \tag{18}$$

The squared norm of the weight error is the sum of first $N$ diagonal elements of $\overline{C}$. In coordinates for which $R$ is diagonal and with $\mu(t) = \mu_0/t$, we find that for $t \gg t_0$

$$E[|v|^2] \approx \sum_{i=1}^{N} \left\{ \overline{C}_{ii}(t_0) \left(\frac{t_0}{t}\right)^{\frac{2\mu_0 \lambda_i}{1-\beta}} + \right.$$

$$\left. \frac{\mu_0^2\, D_{ii}}{(1-\beta)\,(2\mu_0 \lambda_i - 1 + \beta)} \left( \frac{1}{t} - \frac{1}{t_0} \left(\frac{t_0}{t}\right)^{\frac{2\mu_0 \lambda_i}{1-\beta}} \right) \right\}. \tag{19}$$

This reduces to (9) when $\beta = 0$. Equation (19) defines two regimes of interest:

1. $\mu_0/(1-\beta) > \mu_{crit}$: $E[|v|^2]$ drops off asymptotically as $1/t$.
2. $\mu_0/(1-\beta) < \mu_{crit}$: $E[|v|^2]$ drops off asymptotically as

$$\left(\frac{1}{t}\right)^{\frac{2\mu_0 \lambda_{min}}{1-\beta}},$$

i.e. *more slowly than $1/t$.*

The form of (19) and the conditions following it show that the asymptotics of gradient descent with momentum are governed by the *effective learning rate*

$$\mu_{eff} \equiv \frac{\mu}{1-\beta} .$$

Figure 2 compares simulations with the predictions of (19) for fixed $\mu_0$ and various $\beta$. The simulations were performed on an ensemble of 2000 networks trained by LMS as described previously but with an additional momentum term of the form given in (11). The upper three curves (dotted) show the behavior of $E[|v|^2]$ for $\mu_{eff} < \mu_{crit}$. The solid curves show the behavior for $\mu_{eff} > \mu_{crit}$.

The derivation of asymptotic normality proceeds similarly to the case without momentum. Again the reader is referred to (Orr and Leen, 1994) for details.

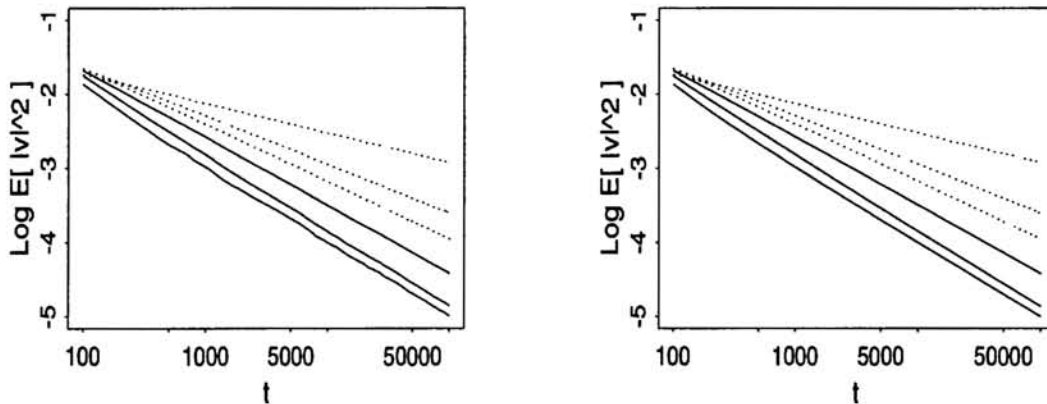

Fig.2: LEFT – Simulation results from an ensemble of 2000 one-dimensional LMS algorithms with momentum with $R = 1.0$, $\sigma^2 = 1.0$, and $\mu_0 = 0.2$.   RIGHT – Theoretical predictions from equation (19). Curves correspond to (top to bottom) $\beta = 0.0, 0.4, 0.5, 0.6, 0.7, 0.8$ .

## 4   Adaptive Momentum Insures Optimal Convergence

The optimal constant momentum parameter is obtained by minimizing the coefficient of $1/t$ in (19). Imposing the restriction that this parameter is positive[3] gives

$$\beta_{opt} = \max(0, 1 - \mu_0 \lambda_{min}). \tag{20}$$

As with $\mu_{opt}$, this result is not of practical use because, in general, $\lambda_{min}$ is unknown.

For 1-dimensional linear networks, an alternative is to use the instantaneous estimate of $\lambda$, $\widehat{\lambda}(t) = x^2(t)$ where $x(t)$ is the network input at time $t$. We thus define the *adaptive momentum parameter* to be

$$\beta_{adapt} = \max(0, 1 - \mu_0 x^2) \quad \text{(1-dimension)}. \tag{21}$$

An algorithm based on (21) insures that the late time convergence is optimally fast. An alternative route to achieving the same goal is to dispense with the momentum term and adaptively adjust the learning rate. Vetner (1967) proposed an algorithm

that iteratively estimates $\lambda$ for 1-D algorithms and uses the estimate to adjust $\mu_0$. Darken and Moody (1992) propose measuring an auxiliary statistic they call "drift" that is used to determine whether or not $\mu_0 > \mu_{crit}$. The adaptive momentum scheme generalizes to multiple dimensions more easily than Vetner's algorithm, and, unlike Darken and Moody's scheme, does not involve calculating an auxiliary statistic not directly involved with the minimization.

A natural extension to N dimensions is to define a matrix of momentum coefficients, $\gamma = I - \mu_0\, x\, x^T$, where $I$ is the $N \times N$ identity matrix. By zeroing out the negative eigenvalues of $\gamma$, we obtain the *adaptive momentum matrix*

$$\beta_{adapt} = I - c\,x\,x^T, \qquad \text{where } c = \min(\mu_0, 1/(x^T x)). \qquad (22)$$

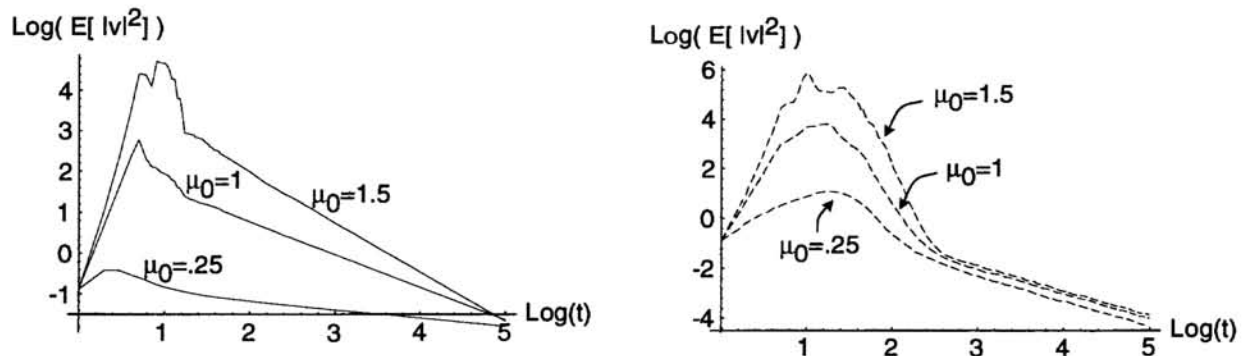

Fig.3: Simulations of 2-D LMS with 1000 networks initialized at $v_0 = (.2, .3)$ and with $\sigma^2 = 1$, $\lambda_1 = .4$, $\lambda_2 = 4$, and $\mu_{crit} = 1.25$. LEFT- $\beta = 0$, RIGHT - $\beta = \beta_{adapt}$. Dashed curves correspond to adaptive momentum.

Figure 3 shows that our adaptive momentum not only achieves the optimal convergence rate *independent of the learning rate parameter* $\mu_0$ but that the value of $\log(E[|v|^2])$ at late times is nearly independent of $\mu_0$ and smaller than when momentum is not used. The left graph displays simulation results without momentum. Here, convergence rates clearly depend on $\mu_0$ and are optimal for $\mu_0 > \mu_{crit} = 1.25$. When $\mu_0$ is large there is initially significant spreading in $v$ so that the increased convergence rate does not result in lower $\log(E[|v|^2])$ until very late times ($t \gtrsim 10^5$). The graph on the right shows simulations with adaptive momentum. Initially, the spreading is even greater than with no momentum, but $\log(E[|v|^2])$ quickly decreases to reach a much smaller value. In addition, for $t \gtrsim 300$, the optimal convergence rate (slope=-1) is achieved for all three values of $\mu_0$ and the curves themselves lie almost on top of one another. In other words, at late times ($t \gtrsim 300$), the value of $\log(E[|v|^2])$ is *independent* of $\mu_0$ when adaptive momentum is used.

## 5  Summary

We have used the dynamics of the weight space probabilities to derive the asymptotic behavior of the weight error correlation for annealed stochastic gradient algorithms with momentum. The late time behavior is governed by the *effective* learning rate $\mu_{eff} \equiv \mu_0/(1 - \beta)$. For learning rate schedules $\mu_0/t$, if $\mu_{eff} > 1/(2\,\lambda_{min})$, then the squared norm of the weight error $v \equiv \omega - \omega_*$ falls off as $1/t$. From these results we have developed a form of momentum that adapts to obtain optimal convergence rates *independent* of the learning rate parameter.

## Acknowledgments

This work was supported by grants from the Air Force Office of Scientific Research (F49620-93-1-0253) and the Electric Power Research Institute (RP8015-2).

## Footnotes

[1]Although algorithms are executed in discrete time, continuous time formulations are often advantagous for analysis. The passage from discrete to continuous time is treated in various ways depending on the needs of the theoretical exposition. Kushner and Clark (1978) define continous time functions that interpolate the discrete time process in order to establish an equivalence between the asymptotic behavior of the discrete time stochastic process, and solutions of an associated deterministic differential equation. Heskes *et al.* (1992) draws on the results of Bedeaux *et al.* (1971) that link (discrete time) random walk trajectories to the solution of a (continuous time) master equation. Heskes' master equation is equivalent to our Kramers-Moyal expansion (3).

[2] In general the density will have nonzero components outside the basin of $\omega_*$. We are neglecting these, for the purpose of calculating the second moment of the the *local* density in the vicinity of $\omega_*$.

[3] $E[|v|^2]$ diverges for $|\beta| > 1$. For $-1 < \beta < 0$, $E[|v|^2]$ appears to converge but oscillations are observed. Additional study is required to determine whether $\beta$ in this range might be useful for improving learning.

## References

D. Bedeaux, K. Laktos-Lindenberg, and K. Shuler. (1971) On the Relation Between Master Equations and Random Walks and their Solutions. *Journal of Mathematical Physics*, 12:2116-2123.

Christian Darken and John Moody. (1992) Towards Faster Stochastic Gradient Search. In J.E. Moody, S.J. Hanson, and R.P. Lipmann (eds.) *Advances in Neural Information Processing Systems, vol. 4*. Morgan Kaufmann Publishers, San Mateo, CA, 1009-1016.

Larry Goldstein. (1987) Mean Square Optimality in the Continuous Time Robbins Monro Procedure. Technical Report DRB-306, Dept. of Mathematics, University of Southern California, LA.

H.J. Kushner and D.S. Clark. (1978) *Stochastic Approximation Methods for Constrained and Unconstrained Systems*. Springer-Verlag, New York.

Tom M. Heskes, Eddy T.P. Slijpen, and Bert Kappen. (1992) Learning in Neural Networks with Local Minima. *Physical Review A*, 46(8):5221-5231.

Todd K. Leen and John E. Moody. (1993) Weight Space Probability Densities in Stochastic Learning: I. Dynamics and Equilibria. In Giles, Hanson, and Cowan (eds.), *Advances in Neural Information Processing Systems, vol. 5*, Morgan Kaufmann Publishers, San Mateo, CA, 451-458.

G. B. Orr and T. K. Leen. (1993) Weight Space Probability Densities in Stochastic Learning: II. Transients and Basin Hopping Times. In Giles, Hanson, and Cowan (eds.), *Advances in Neural Information Processing Systems, vol. 5*, Morgan Kaufmann Publishers, San Mateo, CA, 507-514.

G. B. Orr and T. K. Leen. (1994) Momentum and Optimal Stochastic Search. In M. C. Mozer, P. Smolensky, D. S. Touretzky, J. L. Elman, and A. S. Weigend (eds.), *Proceedings of the 1993 Connectionist Models Summer School*, 351-357.

John J. Shynk and Sumit Roy. (1988) The LMS Algorithm with Momentum Updating. *Proceedings of the IEEE International Symposium on Circuits and Systems*, 2651-2654.

Mehmet Ali Tugay and Yalçin Tanik. (1989) Properties of the Momentum LMS Algorithm. *Signal Processing*, 18:117-127.

J. H. Venter. (1967) An Extension of the Robbins-Monro Procedure. *Annals of Mathematical Statistics*, 38:181-190.

Halbert White. (1989) Learning in Artificial Neural Networks: A Statistical Perspective. *Neural Computation*, 1:425-464.